# Sequential Adaptation of Radial Basis Function Neural Networks and its Application to Time-series Prediction

**V. Kadirkamanathan**
Engineering Department
Cambridge University
Cambridge CB2 1PZ, UK

**M. Niranjan**

**F. Fallside**

## Abstract

We develop a sequential adaptation algorithm for radial basis function (RBF) neural networks of Gaussian nodes, based on the method of successive $\mathcal{F}$-Projections. This method makes use of each observation efficiently in that the network mapping function so obtained is consistent with that information and is also optimal in the least $L_2$-norm sense. The RBF network with the $\mathcal{F}$-Projections adaptation algorithm was used for predicting a chaotic time-series. We compare its performance to an adaptation scheme based on the method of stochastic approximation, and show that the $\mathcal{F}$-Projections algorithm converges to the underlying model much faster.

## 1 INTRODUCTION

Sequential adaptation is important for signal processing applications such as time-series prediction and adaptive control in nonstationary environments. With increasing computational power, complex algorithms that can offer better performance can be used for these tasks. A sequential adaptation scheme, called the method of successive $\mathcal{F}$-Projections [Kadirkamanathan & Fallside, 1990], makes use of each observation efficiently in that, the function so obtained is consistent with that observation and is the optimal posterior in the least $L_2$-norm sense.

In this paper we present an adaptation algorithm based on this method for the radial basis function (RBF) network of Gaussian nodes [Broomhead & Lowe, 1988]. It is a memoryless adaptation scheme since neither the information about the past samples nor the previous adaptation directions are retained. Also, the observations are presented only once. The RBF network employing this adaptation scheme

was used for predicting a chaotic time-series. The performance of the algorithm is compared to a memoryless sequential adaptation scheme based on the method of stochastic approximation.

## 2   METHOD OF SUCCESSIVE $\mathcal{F}$-PROJECTIONS

The *principle of $\mathcal{F}$-Projection* [Kadirkamanathan *et al.*, 1990] is a general method of choosing a posterior function estimate of an unknown function $f^*$, when there exists a prior estimate and new information about $f^*$ in the form of constraints. The principle states that, of all the functions that satisfy the constraints, one should choose the posterior $f_n$ that has the least $L_2$-norm, $\|f_n - f_{n-1}\|$, where $f_{n-1}$ is the prior estimate of $f^*$. *viz.*,

$$f_n = \arg\min_f \|f - f_{n-1}\| \qquad \text{such that} \qquad f_n \in H_I \qquad (1)$$

where $H_I$ is the set of functions that satisfy the new constraints, and

$$\|f - f_{n-1}\|^2 = \int_{\underline{x} \in C} \|f(\underline{x}) - f_{n-1}(\underline{x})\|^2 |d\underline{x}| = D(f, f_{n-1}) \qquad (2)$$

where $\underline{x}$ is the input vector, $|d\underline{x}|$ is the infinitesimal volume in the input space domain C.

In functional analysis theory, the metric $D(.,.)$ describes the $L_2$-normed linear space of square integrable functions. Since an inner product can be defined in this space, it is also the Hilbert space of square integrable functions [Linz, 1984]. Constraints of the form $y_n = f(\underline{x}_n)$ are linear in this space, and the functions that satisfy the constraint lie in a hyperplane subspace $H_I$. The posterior $f_n$, obtained from the principle can be seen to be a projection of $f_{n-1}$ onto the subspace $H_I$ containing $f^*$, the underlying function that generates the observation set, and hence is optimal (i.e., best possible choice), see Figure 1.

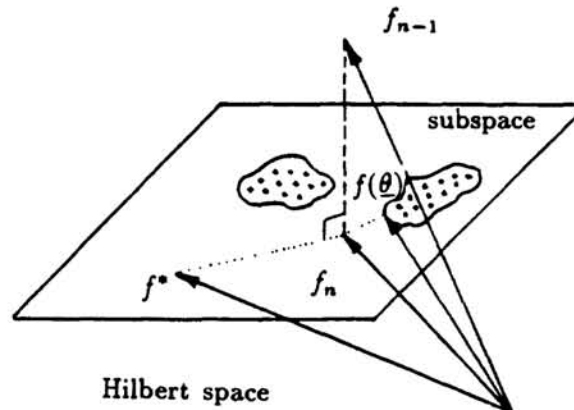

Figure 1: Principle of $\mathcal{F}$-Projection

Neural networks can be viewed as constructing a function in its input space. The structure of the neural network and the finite number of parameters restrict the class

of functions that can be constructed to a subset of functions in the Hilbert space. Neural networks, therefore approximate the underlying function that describes the set of observations. Hence, the principle of $\mathcal{F}$-Projection now yields a posterior $f(\underline{\theta}_n) \in H_I$ that is an approximation of $f_n$ (see Figure 1).

The *method of successive $\mathcal{F}$-Projections* is the application of the principle of $\mathcal{F}$-Projection on a sequence of observations or information [Kadirkamanathan *et al.*, 1990]. For neural networks, the method gives an algorithm that has the following two steps.

- Initialise parameters with random values or values based on a priori knowledge.
- For each pattern $(\underline{x}_i, y_i)$ $i = 1 \dots n$, determine the posterior parameter estimate

$$\underline{\theta}_i = \arg\min_{\underline{\theta}} \int_{\underline{x} \in C} \| f(\underline{x}, \underline{\theta}) - f(\underline{x}, \underline{\theta}_{i-1}) \|^2 |d\underline{x}|$$

such that $f(\underline{x}_i, \underline{\theta}_i) = y_i$.

where $(\underline{x}_i, y_i)$, for $i = 1 \dots n$ constitutes the observation set, $\underline{\theta}$ is the neural network parameter set and $f(\underline{x}, \underline{\theta})$ is the function constructed by the neural network.

## 3 $\mathcal{F}$-PROJECTIONS FOR AN RBF NETWORK

The class of radial basis function (RBF) neural networks were first introduced by Broomhead & Lowe [1988]. One such network is the RBF network of Gaussian nodes. The function constructed at the output node of the RBF network of Gaussian nodes, $f(\underline{x})$, is derived from a set of basis functions of the form,

$$\phi_i(\underline{x}) = \exp\{-(\underline{x} - \underline{\mu}_i)^T C_i^{-1} (\underline{x} - \underline{\mu}_i)\} \qquad i = 1 \dots m \tag{3}$$

Each basis function $\phi_i(\underline{x})$ is described at the output of each hidden node and is centered on $\underline{\mu}_i$ in the input space. $\phi_i(\underline{x})$ is a function of the radial weighted distance between the input vector $\underline{x}$ and the node centre $\underline{\mu}_i$. In general, $C_i$ is diagonal with elements $[\sigma_{i1}, \sigma I2, \dots, \sigma_{iN}]$. $f(\underline{x})$ is a linear combination of the $m$ basis functions.

$$f(\underline{x}) = \sum_{i=1}^{m} \alpha_i \phi_i(\underline{x}) \tag{4}$$

and $\underline{\theta} = [\cdots, \alpha_i, \underline{\mu}_i, \underline{\sigma}_i, \cdots]$ is then the parameter vector for the RBF network.

There are two reasons for developing the sequential adaptation algorithm for the RBF network of Gaussian nodes. Firstly, the method of successive $\mathcal{F}$-Projections is based on minimizing the hypervolume change in the hypersurface when learning new patterns. The RBF network of Gaussian nodes construct a localized hypersurface and therefore the changes will also be local. This results in the adaptation of a few nodes and therefore the algorithm is quite stable. Secondly, the $L_2$-norm measure of the hypervolume change can be solved analytically for the RBF network of Gaussian nodes.

The method of successive $\mathcal{F}$-Projections is developed under deterministic noise-free conditions. When the observations are noisy, the constraint that $f(\underline{\theta}_i, \underline{x}_i) = y_i$ must be relaxed to,

$$\|f(\underline{\theta}_i, \underline{x}) - y_i\|^2 \leq \epsilon \tag{5}$$

Hence, the sequential adaptation scheme is modified to,

$$\underline{\theta}_n = \arg \min_{\underline{\theta}} J(\underline{\theta}) \tag{6}$$

$$J(\underline{\theta}) = \int \|f(\underline{\theta}, \underline{x}) - f(\underline{\theta}_{i-1}, \underline{x})\|^2 |d\underline{x}| \quad + \quad c_i \|f(\underline{\theta}, \underline{x}_i) - y_i\|^2 \tag{7}$$

$c_i$ is the penalty parameter that trades off between the importance of learning the new pattern and losing the information of the past patterns. This minimization can be performed by the gradient descent procedure. The minimization procedure is halted when the change $\Delta J$ falls below a threshold. The complete adaptation algorithm is as follows:

- Choose $\underline{\theta}_0$ randomly
- For each pattern $(i = 1 \ldots P)$
  - $\underline{\theta}_i^{(0)} = \underline{\theta}_{i-1}$
  - Repeat ($k^{th}$ iteration)

    $$\underline{\theta}_i^{(k)} = \underline{\theta}_i^{(k-1)} - \eta \nabla J \Big|_{\underline{\theta} = \underline{\theta}_i^{(k-1)}}$$

    Until $\Delta J^{(k)} < \varepsilon_{th}$

where $\nabla J$ is the gradient vector of $J(\underline{\theta})$ with respect to $\underline{\theta}$, $\Delta J^{(k)} = J(\underline{\theta}_i^{(k)}) - J(\underline{\theta}_i^{(k-1)})$ is the change in the cost function and $\varepsilon_{th}$ is a threshold. Note that $\alpha_i, \underline{\mu}_i, \underline{\sigma}_i$ for $i = 1 \ldots m$ are all adapted. The details of the algorithm can be found in the report by Kadirkamanathan et al., [Kadirkamanathan, Niranjan & Fallside, 1991].

## 4    TIME SERIES PREDICTION

An area of application for sequential adaptation of neural networks is the prediction of time-series in nonstationary environments, where the underlying model generating the time-series is time-varying. The adaptation algorithm must also result in the convergence of the neural network to the underlying model under stationary conditions. The usual approach to predicting time-series is to train the neural network on a set of training data obtained from the series [Lapedes & Farber, 1987; Farmer & Sidorowich, 1988; Niranjan, 1991]. Our sequential adaptation approach differs from this in that the adaptation takes place for each sample.

In this work, we examine the performance of the $\mathcal{F}$-Projections adaptation algorithm for the RBF network of Gaussian nodes in predicting a deterministic chaotic series. The chaotic series under investigation is the *logistic map* [Lapedes & Farber, 1987], whose dynamics is governed by the equation,

$$x_n = 4x_{n-1}(1 - x_{n-1}) \tag{8}$$

This is a first order nonlinear process where only the previous sample determines the value of the present sample. Since neural networks offer the capability of constructing any arbitrary mapping to a sufficient accuracy, a network with input nodes equal to the process order will find the underlying model. Hence, we use the RBF network of Gaussian nodes with a single input node. We are thus able to compare the map the RBF network constructed with that of the actual map given by eqn (8).

First, RBF network with 2 input nodes and 8 Gaussian nodes was used to predict the logistic map chaotic series of 100 samples. Each sample was presented only once for training. The training was temporarily halted after 0, 20, 40, 60, 80 and 100 samples, and in each case the prediction error residual was found. This is given in Figure 2 where the increasing darkness of the curves stand for the increasing number of patterns used for training. It is evident from this figure that the prediction model improves very quickly from the initial state and then slowly keeps on improving as the number of training patterns used is increased.

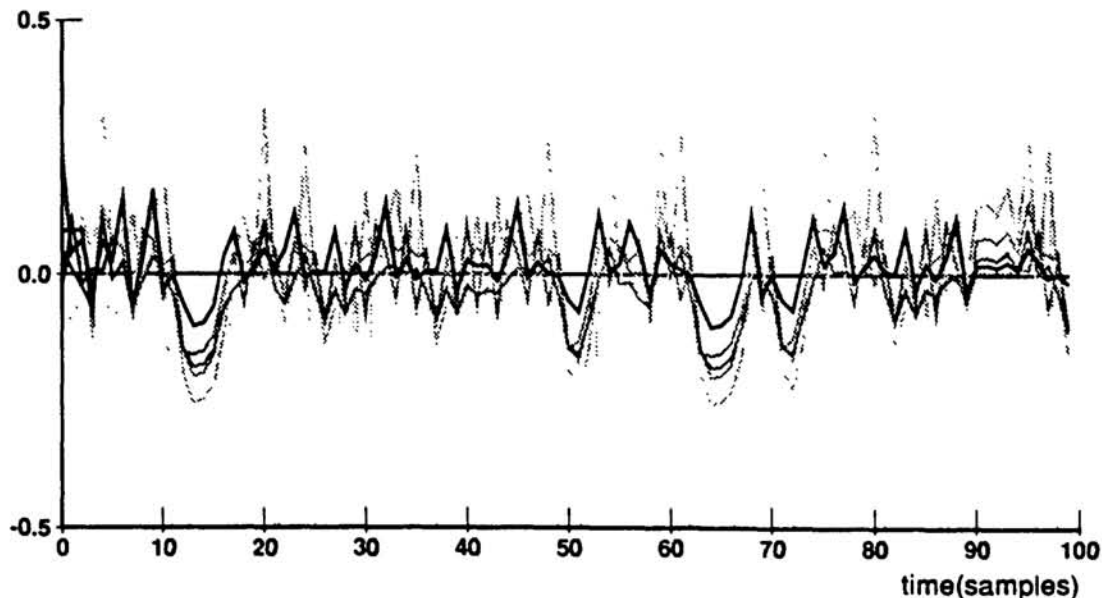

Figure 2: Evolution of prediction error residuals

In order to compare the performance of the sequential adaptation algorithm, a memoryless adaptation scheme was also used to predict the chaotic series. The scheme is the LMS or stochastic approximation (sequential back propagation [White, 1987]), where for each sample, one iteration takes place. The iteration is given by,

$$\underline{\theta}_i = \underline{\theta}_{i-1} - \eta \nabla J \Big|_{\underline{\theta} = \underline{\theta}_i} \tag{9}$$

where,

$$J(\underline{\theta}) = \|f(\underline{\theta}, \underline{x}_i) - y_i\|^2 \tag{10}$$

and $J(\underline{\theta})$ is the squared prediction error for the present sample.

Next, the RBF network with a single input node and 8 Gaussian units was used to predict the chaotic series. The $\mathcal{F}$-Projections and the stochastic approximation

adaptation algorithms were used for training this network on 60 samples. Results on the map constructed by a network trained by each of these schemes for 0, 20 and 60 samples and the samples used for training are shown in Figure 3. Again, each sample was presented only once for training.

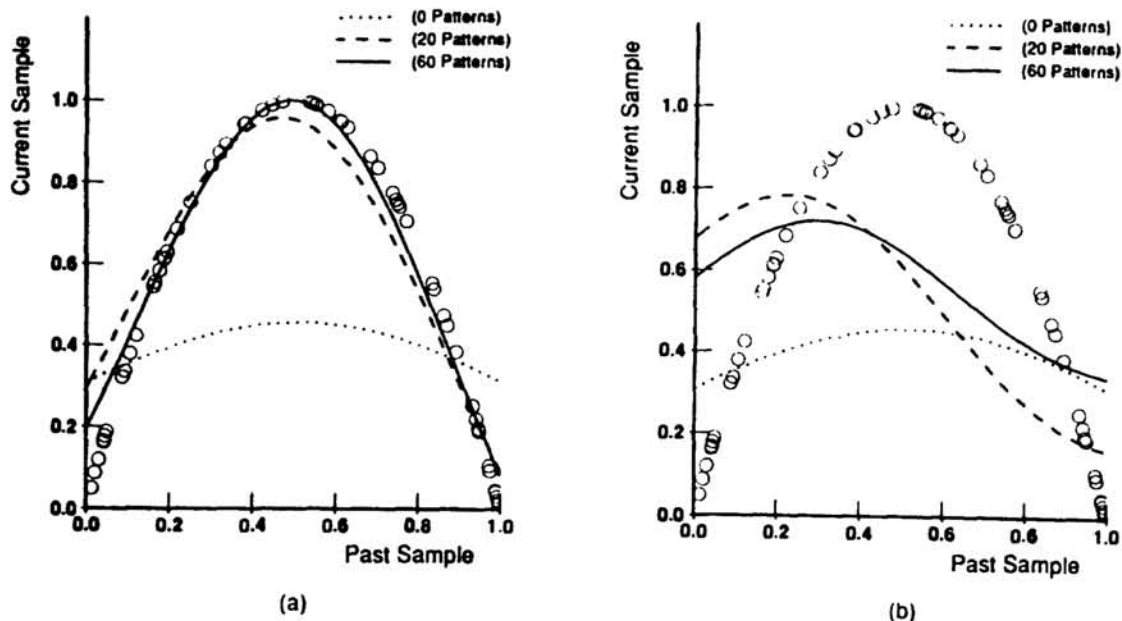

Figure 3: Map f(x) constructed by the RBF network. (a) f-projections (b) stochastic approximation.

The stochastic approximation algorithm fails to construct a close-fit mapping of the underlying function after training on 60 samples. The $\mathcal{F}$-Projections algorithm however, provides a close-fit map after training on 20 samples. It also shows stability by maintaining the map up to training on 60 samples. The speed of convergence achieved, in terms of the number of samples used for training, is much higher for the $\mathcal{F}$-Projections.

Comparing the cost functions being minimized for the $\mathcal{F}$-Projections and the stochastic approximation algorithms, given by eqns (7) and (10), it is clear that the difference is only an additional integral term in eqn (7). This term is not a function of the present observation, but is a function of the *a priori* parameter values. The addition of such a term is to incorporate a priori knowledge of the network to that of the present observation in determining the posterior parameter values. The faster convergence result for the $\mathcal{F}$-Projections indicate the importance of the extended cost function. Even though the cost term for the $\mathcal{F}$-Projections was developed for a recursive estimation algorithm, it can be applied to a block estimation method as well. The cost function given by eqn (7) can be seen to be an extension of the nonlinear least squared error to incorporate *a priori* knowledge.

## 5   CONCLUSIONS

The *principle of $\mathcal{F}$-Projection* proposed by Kadirkamanathan *et al.*, [1990], provides an optimal posterior estimate of a function, from the prior estimate and new

information. Based on it, they propose a sequential adaptation scheme called, the *method of successive f-projections*. We have developed a sequential adaptation algorithm for the RBF network of Gaussian nodes based on this method.

Applying the RBF network with the $\mathcal{F}$-Projections algorithm to the prediction of a chaotic series, we have found that the RBF network was able to map the underlying function. The prediction error residuals at the end of training with different number of samples, indicate that, after a substantial reduction in the error in the initial stages, with increasing number of samples presented for training the error was steadily decreasing. By comparing with the performance of the stochastic approximation algorithm, we show the superior convergence achieved by the $\mathcal{F}$-Projections.

Comparing the cost functions being minimized for the $\mathcal{F}$-Projections and the stochastic approximation algorithms reveal that the $\mathcal{F}$-Projections uses both the prediction error for the current sample and the *a priori* values of the parameters, whereas the stochastic approximation algorithms use only the prediction error. We also point out that such a cost term that includes a priori knowledge of the network can be used for training a trained network upon receipt of further information.

# References

[1] Broomhead.D.S & Lowe.D, (1988), *"Multi-variable Interpolation and Adaptive Networks"*, RSRE memo No.4148, Royal Signals and Radar Establishment, Malvern.

[2] Farmer.J.D & Sidorowich.J.J, (1988), *"Exploiting chaos to predict the future and reduce noise"*, Technical Report, Los Alamos National Laboratory.

[3] Kadirkamanathan.V & Fallside.F (1990), *"F-Projections: A nonlinear recursive estimation algorithm for neural networks"*, Technical Report CUED/F-INFENG/TR.53, Cambridge University Engineering Department.

[4] Kadirkamanathan.V, Niranjan.M & Fallside.F (1991), *"Adaptive RBF network for time-series prediction"*, Technical Report CUED/F-INFENG/TR.56, Cambridge University Engineering Department.

[5] Lapedes.A.S & Farber.R, (1987), *"Non-linear signal processing using neural networks: Prediction and system modelling"*, Technical report, Los Alamos National Laboratory, Los Alamos, New Mexico 87545.

[6] Linz.P, (1984), *"Theoretical Numerical Analysis"*, John Wiley, New York.

[7] Niranjan.M, (1991), *"Implementing threshold autoregressive models for time series prediction on a multilayer perceptron"*, Technical Report CUED/F-INFENG/TR.50, Cambridge University Engineering Department.

[8] White.H, 1987, *"Some asymptotic results for learning in single hidden layer feedforward network models"*, Technical Report, Department of Economics, Univeristy of California, San Diego.
